# Scalable Training of Mixture Models via Coresets

**Dan Feldman**
MIT

**Matthew Faulkner**
Caltech

**Andreas Krause**
ETH Zurich

## Abstract

How can we train a statistical mixture model on a massive data set? In this paper, we show how to construct *coresets* for mixtures of Gaussians and natural generalizations. A coreset is a weighted subset of the data, which guarantees that models fitting the coreset will also provide a good fit for the original data set. We show that, perhaps surprisingly, Gaussian mixtures admit coresets of size *independent* of the size of the data set. More precisely, we prove that a weighted set of $O(dk^3/\varepsilon^2)$ data points suffices for computing a $(1 + \varepsilon)$-approximation for the optimal model on the original $n$ data points. Moreover, such coresets can be efficiently constructed in a map-reduce style computation, as well as in a streaming setting. Our results rely on a novel reduction of statistical estimation to problems in computational geometry, as well as new complexity results about mixtures of Gaussians. We empirically evaluate our algorithms on several real data sets, including a density estimation problem in the context of earthquake detection using accelerometers in mobile phones.

## 1 Introduction

We consider the problem of training statistical mixture models, in particular mixtures of Gaussians and some natural generalizations, on massive data sets. Such data sets may be distributed across a cluster, or arrive in a data stream, and have to be processed with limited memory. In contrast to parameter estimation for models with compact sufficient statistics, mixture models generally require inference over latent variables, which in turn depends on the full data set. In this paper, we show that Gaussian mixture models (GMMs), and some generalizations, admit small *coresets*: A coreset is a weighted subset of the data which guarantees that models fitting the coreset will also provide a good fit for the original data set. Perhaps surprisingly, we show that Gaussian mixtures admit coresets of size *independent* of the size of the data set.

We focus on $\varepsilon$-semi-spherical Gaussians, where the covariance matrix $\Sigma_i$ of each component $i$ has eigenvalues bounded in $[\varepsilon, 1/\varepsilon]$, but some of our results generalize even to the semi-definite case. In particular, we show that given a data set $D$ of $n$ points in $\mathbb{R}^d$, $\varepsilon > 0$ and $k \in \mathbb{N}$, how one can efficiently construct a weighted set $C$ of $\mathcal{O}(dk^3/\varepsilon^2)$ points, such that for any mixture of $k$ $\varepsilon$-semi-spherical Gaussians $\theta = [(w_1, \mu_1, \Sigma_1), \ldots, (w_k, \mu_k, \Sigma_k)]$ it holds that the log-likelihood $\ln P(D \mid \theta)$ of $D$ under $\theta$ is approximated by the (properly weighted) log-likelihood $\ln P(C \mid \theta)$ of $C$ under $\theta$ to arbitrary accuracy as $\varepsilon \to 0$. Thus solving the estimation problem on the coreset $C$ (e.g., using weighted variants of the EM algorithm, see Section 3.3) is almost as good as solving the estimation problem on large data set $D$. Our algorithm for constructing $C$ is based on adaptively sampling points from $D$ and is simple to implement. Moreover, coresets can be efficiently constructed in a map-reduce style computation, as well as in a streaming setting (using space and update time per point of $\mathrm{poly}(dk\varepsilon^{-1} \log n \log(1/\delta))$).

Existence and construction of coresets have been investigated for a number of problems in computational geometry (such as $k$-means and $k$-median) in many recent papers (*cf.*, surveys in [1, 2]). In this paper, we demonstrate how these techniques from computational geometry can be lifted to the realm of statistical estimation. As a by-product of our analysis, we also close an open question on the VC dimension of arbitrary mixtures of Gaussians. We evaluate our algorithms on several synthetic and real data sets. In particular, we use our approach for density estimation for acceleration data, motivated by an application in earthquake detection using mobile phones.

## 2    Background and Problem Statement

**Fitting mixture models by MLE.**    Suppose we are given a data set $D = \{\mathbf{x}_1, \ldots, \mathbf{x}_n\} \subseteq \mathbb{R}^d$. We consider fitting a mixture of Gaussians $\theta = [(w_1, \mu_1, \Sigma_1), \ldots, (w_k, \mu_k, \Sigma_k)]$, i.e., the distribution $P(\mathbf{x} \mid \theta) = \sum_{i=1}^{k} w_i \mathcal{N}(\mathbf{x}; \mu_i, \Sigma_i)$, where $w_1, \ldots, w_k \geq 0$ are the mixture weights, $\sum_i w_i = 1$, and $\mu_i$ and $\Sigma_i$ are mean and covariance of the $i$-th mixture component, which is modeled as a multivariate normal distribution $\mathcal{N}(\mathbf{x}, \mu_i, \Sigma_i) = \frac{1}{\sqrt{|2\pi\Sigma_i|}} \exp\left(-\frac{1}{2}(\mathbf{x} - \mu_i)^T \Sigma_i^{-1}(\mathbf{x} - \mu_i)\right)$. In Section 4, we will discuss extensions to more general mixture models. Assuming the data was generated i.i.d., the negative log likelihood of the data is $\mathcal{L}(D \mid \theta) = -\sum_j \ln P(\mathbf{x}_j \mid \theta)$, and we wish to obtain the maximum likelihood estimate (MLE) of the parameters $\theta^* = \operatorname{argmin}_{\theta \in \mathfrak{C}} \mathcal{L}(D \mid \theta)$, where $\mathfrak{C}$ is a set of constraints ensuring that degenerate solutions are avoided[1]. Hereby, for a symmetric matrix $\mathbf{A}$, spec $\mathbf{A}$ is the set of all eigenvalues of $\mathbf{A}$. We define

$$\mathfrak{C} = \mathfrak{C}_\varepsilon = \{\theta = [(w_1, \mu_1, \Sigma_1), \ldots, (w_k, \mu_k, \Sigma_k)] \mid \forall_i : \operatorname{spec}(\Sigma_i) \subseteq [\varepsilon, 1/\varepsilon]\}$$

to be the set of all mixtures of $k$ Gaussians $\theta$, such that all the eigenvalues of the covariance matrices of $\theta$ are bounded between $\varepsilon$ and $1/\varepsilon$ for some small $\varepsilon > 0$.

**Approximating the log-likelihood.**    Our goal is to approximate the data set $D$ by a weighted set $C = \{(\gamma_1, \mathbf{x}'_1), \ldots, (\gamma_m, \mathbf{x}'_m)\} \subseteq \mathbb{R} \times \mathbb{R}^d$, such that $\mathcal{L}(D \mid \theta) \approx \mathcal{L}(C \mid \theta)$ for all $\theta$, where we define $\mathcal{L}(C \mid \theta) = -\sum_i \gamma_i \ln P(\mathbf{x}'_i \mid \theta)$.

What kind of approximation accuracy may we hope to expect? Notice that there is a nontrivial issue of scale: Suppose we have a MLE $\theta^*$ for $D$, and let $\alpha > 0$. Then straightforward linear algebra shows that we can obtain an MLE $\theta^*_\alpha$ for a scaled data set $\alpha D = \{\alpha \mathbf{x} : \mathbf{x} \in D\}$ by simply scaling all means by $\alpha$, and covariance matrices by $\alpha^2$. For the log-likelihood, however, it holds that $\mathcal{L}(\alpha D \mid \theta^*_\alpha) = d \ln \alpha + \mathcal{L}(D \mid \theta^*)$. Therefore, optimal solutions on one scale can be efficiently transformed to optimal solutions at a different scale, while maintaining the same *additive error*. This means, that any algorithm which achieves absolute error $\varepsilon$ at any scale could be used to achieve parameter estimates (for means, covariances) with *arbitrarily* small error, simply by applying the algorithm to a scaled data set and transforming back the obtained solution. An alternative, scale-invariant approach may be to strive towards approximating $\mathcal{L}(D \mid \theta)$ up to *multiplicative error* $(1 + \varepsilon)$. Unfortunately, this goal is also hard to achieve: Choosing a scaling parameter $\alpha$ such that $d \ln \alpha + \mathcal{L}(D \mid \theta^*) = 0$ would require any algorithm that achieves any bounded multiplicative error to essentially incur *no error at all* when evaluating $\mathcal{L}(\alpha D \mid \theta^*)$. The above observations hold even for the case $k = 1$ and $\Sigma = I$, where the mixture $\theta$ consists of a single Gaussian, and the log-likelihood is the sum of squared distances to a point $\mu$ and an additive term.

Motivated by the scaling issues discussed above, we use the following error bound that was suggested in [3] (who studied the case where all Gaussians are identical spheres). We decompose the negative log-likelihood $\mathcal{L}(D \mid \theta)$ of a data set $D$ as

$$\mathcal{L}(D \mid \theta) = -\sum_{j=1}^{n} \ln \sum_{i=1}^{k} \frac{w_i}{\sqrt{|2\pi\Sigma_i|}} \exp\left(-\frac{1}{2}(\mathbf{x}_j - \mu_i)^T \Sigma_i^{-1}(\mathbf{x}_j - \mu_i)\right) = -n \ln Z(\theta) + \phi(D \mid \theta)$$

where $Z(\theta) = \sum_i \frac{w_i}{\sqrt{|2\pi\Sigma_i|}}$ is a normalizer, and the function $\phi$ is defined as

$$\phi(D \mid \theta) = -\sum_{j=1}^{n} \ln \sum_{i=1}^{k} \frac{w_i}{Z(\theta)\sqrt{|2\pi\Sigma_i|}} \exp\left(-\frac{1}{2}(\mathbf{x}_j - \mu_i)^T \Sigma_i^{-1}(\mathbf{x}_j - \mu_i)\right).$$

Hereby, $Z(\theta)$ plays the role of a normalizer, which can be computed *exactly*, independently of the set $D$. $\phi(D \mid \theta)$ captures all dependencies of $\mathcal{L}(D \mid \theta)$ on $D$, and via Jensen's inequality, it can be seen that $\phi(D \mid \theta)$ is always nonnegative.

We can now use this term $\phi(D \mid \theta)$ as a reference for our error bounds. In particular, we call $\tilde{\theta}$ a $(1 + \varepsilon)$-approximation for $\theta$ if $(1 - \varepsilon)\phi(D \mid \theta) \leq \phi(D \mid \tilde{\theta}) \leq \phi(D \mid \theta)(1 + \varepsilon)$.

**Coresets.**    We call a weighted data set $C$ a $(k, \varepsilon)$-*coreset* for another (possibly weighted) set $D \subseteq \mathbb{R}^d$, if for all mixtures $\theta \in \mathfrak{C}$ of $k$ Gaussians it holds that

$$(1 - \varepsilon)\phi(D \mid \theta) \leq \phi(C \mid \theta) \leq \phi(D \mid \theta)(1 + \varepsilon).$$

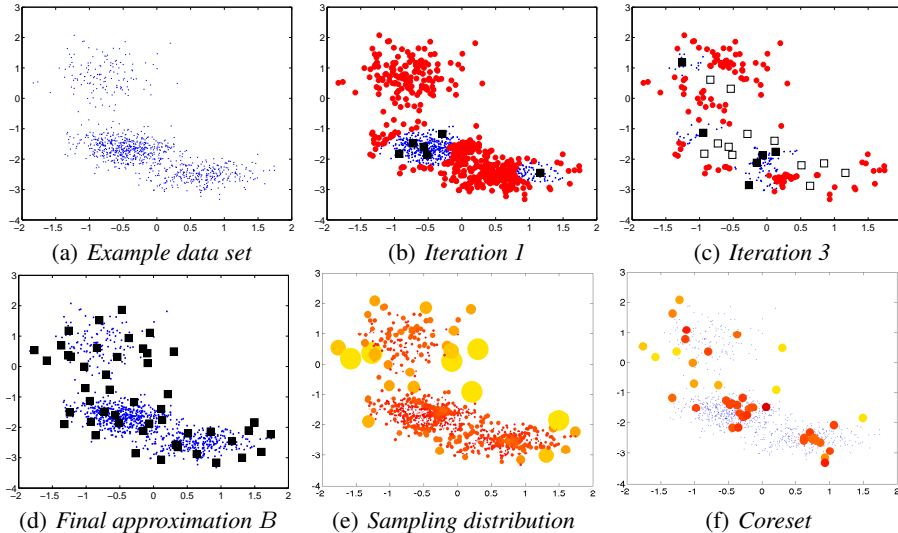

(a) *Example data set*     (b) *Iteration 1*     (c) *Iteration 3*

(d) *Final approximation B*     (e) *Sampling distribution*     (f) *Coreset*

Figure 1: Illustration of the coreset construction for example data set (a). (b,c) show two iterations of constructing the set $B$. Solid squares are points sampled uniformly from remaining points, hollow squares are points selected in previous iterations. Red color indicates half the points furthest away from $B$, which are kept for next iteration. (d) final approximate clustering $B$ on top of original data set. (e) Induced non-uniform sampling distribution: radius of circles indicates probability; color indicates weight, ranging from red (high weight) to yellow (low weight). (f) Coreset sampled from distribution in (e).

Hereby $\phi(C \mid \theta)$ is generalized to weighted data sets $C$ in the natural way (weighing the contribution of each summand $\mathbf{x}'_j \in C$ by $\gamma_j$). Thus, as $\varepsilon \to 0$, for a sequence of $(k, \varepsilon)$-coresets $C_\varepsilon$ we have that $\sup_{\theta \in \mathfrak{C}} |\mathcal{L}(C_\varepsilon \mid \theta) - \mathcal{L}(D \mid \theta)| \to 0$, i.e., $\mathcal{L}(C_\varepsilon \mid \theta)$ uniformly (over $\theta \in \mathfrak{C}$) approximates $\mathcal{L}(D \mid \theta)$. Further, under the additional condition that all variances are sufficiently large (formally $\prod_{\lambda \in \mathrm{spec}(\Sigma_i)} \lambda \geq \frac{1}{(2\pi)^d}$ for all components $i$), the log-normalizer $\ln Z(\theta)$ is negative, and consequently the coreset in fact provides a multiplicative $(1 + \varepsilon)$ approximation to the log-likelihood, i.e.,

$$(1 - \varepsilon)\mathcal{L}(D \mid \theta) \leq \mathcal{L}(C \mid \theta) \leq \mathcal{L}(D \mid \theta)(1 + \varepsilon).$$

More details can be found in the supplemental material.

Note that if we had access to a $(k, \varepsilon)$-coreset $C$, then we could reduce the problem of fitting a mixture model on $D$ to one of fitting a model on $C$, since the optimal solution $\theta_C$ is a good approximation (in terms of log-likelihood) of $\theta^*$. While finding the optimal $\theta_C$ is a difficult problem, one can use a (weighted) variant of the EM algorithm to find a good solution. Moreover, if $|C| \ll |D|$, running EM on $C$ may be orders of magnitude faster than solving it on $D$. In Section 3.3, we give more details about solving the density estimation problem on the coreset.

The key question is whether small $(k, \varepsilon)$-coresets exist, and whether they can be efficiently constructed. In the following, we answer this question affirmatively. We show that, perhaps surprisingly, one can efficiently find coresets $C$ of size *independent* of the size $n$ of $D$, and with polynomial dependence on $\frac{1}{\varepsilon}$, $d$ and $k$.

## 3 Efficient Coreset Construction via Adaptive Sampling

**Naive approach: uniform sampling.** A naive approach towards approximating $D$ would be to just pick a subset $C$ uniformly at random. In particular, suppose the data set is generated from a mixture of two spherical Gaussians ($\Sigma_i = \mathbf{I}$) with weights $w_1 = \frac{1}{\sqrt{n}}$ and $w_2 = 1 - \frac{1}{\sqrt{n}}$. Unless $m = \Omega(\sqrt{n})$ points are sampled, with constant probability no data point generated from Gaussian 2 is selected. By moving the means of the Gaussians arbitrarily far apart, $\mathcal{L}(D \mid \theta_C)$ can be made arbitrarily worse than $\mathcal{L}(D \mid \theta_D)$, where $\theta_C$ and $\theta_D$ are MLEs on $C$ and $D$ respectively. Thus, even for two well-separated Gaussians, uniform sampling can perform arbitrarily poorly. This example already suggests that, intuitively, in order to achieve small multiplicative error, we must devise a sampling scheme that adaptively selects representative points from all "clusters" present in the data set. However, this suggests that obtaining a coreset requires solving a chicken-and-egg problem, where we need to understand the density of the data to obtain the coreset, but simultaneously would like to use the coreset for density estimation.

**Better approximation via adaptive sampling.** The key idea behind the coreset construction is that we can break the chicken-and-egg problem by first obtaining a rough approximation $B$ of the clustering solution (using more than $k$ components, but far fewer than $n$), and then to use this solution to bias the random sampling. Surprisingly, a simple procedure which iteratively samples a small number $\beta$ of points, and removes half of the data set closest to the sampled points, provides a sufficiently accurate first approximation $B$ for this purpose. This initial clustering is then used to sample the data points comprising coreset $C$ according to probabilities which are roughly proportional to the squared distance to the set $B$. This non-uniform random sampling can be understood as an importance-weighted estimate of the log-likelihood $\mathcal{L}(D \mid \theta)$, where the weights are optimized in order to reduce the variance. The same general idea has been found successful in constructing coresets for geometric clustering problems such as $k$-means and $k$-median [4]. The pseudocode for obtaining the approximation $B$, and for using it to obtain coreset $C$ is given in Algorithm 1.

---

**Algorithm 1**: Coreset construction

---

**Input**: Data set $D$, $\varepsilon$, $\delta$, $k$
**Output**: Coreset $C = \big\{ (\gamma(\mathbf{x}_1), \mathbf{x}_1), \ldots, (\gamma(\mathbf{x}_{|C|}), \mathbf{x}_{|C|}) \big\}$
$D' \leftarrow D$; $B \leftarrow \emptyset$;
**while** $|D'| > 10dk\ln(1/\delta)$ **do**
    Sample set $S$ of $\beta = 10dk\ln(1/\delta)$ points uniformly at random from $D'$;
    Remove $\lceil |D'|/2 \rceil$ points $\mathbf{x} \in D'$ closest to $S$ (i.e., minimizing $\mathrm{dist}(\mathbf{x}, S)$) from $D'$;
    Set $B \leftarrow B \cup S$;
Set $B \leftarrow B \cup D'$;
**for** *each $b \in B$* **do** $D_b \leftarrow$ the points in $D$ whose closest point in $B$ is $b$. Ties broken arbitrarily;
**for** *each $b \in B$ and $\mathbf{x} \in D_b$* **do**
$$m(\mathbf{x}) \leftarrow \left\lceil \frac{5}{|D_b|} + \frac{\mathrm{dist}(\mathbf{x}, B)^2}{\sum_{\mathbf{x}' \in D} \mathrm{dist}(\mathbf{x}', B)^2} \right\rceil;$$
Pick a non-uniform random sample $C$ of $10\lceil dk|B|^2 \ln(1/\delta)/\varepsilon^2 \rceil$ points from $D$, where for every $\mathbf{x}' \in C$ and $\mathbf{x} \in D$, we have $\mathbf{x}' = \mathbf{x}$ with probability $m(\mathbf{x})/\sum_{\mathbf{x}' \in D} m(\mathbf{x}')$;
**for** *each $\mathbf{x}' \in C$* **do** $\gamma(\mathbf{x}') \leftarrow \frac{\sum_{\mathbf{x} \in D} m(\mathbf{x})}{|C| \cdot m(\mathbf{x}')}$;

---

We have the following result, proved in the supplemental material:

**Theorem 3.1.** *Suppose $C$ is sampled from $D$ using Algorithm 1 for parameters $\varepsilon, \delta$ and $k$. Then, with probability at least $1 - \delta$ it holds that for all $\theta \in \mathfrak{C}_\varepsilon$,*
$$\phi(D \mid \theta)(1 - \varepsilon) \leq \phi(C \mid \theta) \leq \phi(D \mid \theta)(1 + \varepsilon).$$

In our experiments, we compare the performance of clustering on coresets constructed via adaptive sampling, vs. clustering on a uniform sample. The size of $C$ in Algorithm 1 depends on $|B|^2 = \log^2 n$. By replacing $B$ in the algorithm with a constant factor approximation $B'$, $|B'| = l$ for the $k$-means problem, we can get a coreset $C$ of size *independent* of $n$. Such a set $B'$ can be computed in $O(ndk)$ time either by applying exhaustive search on the output $C$ of the original Algorithm 1 or by using one of the existing constant-factor approximation algorithms for $k$-means (say, [5]).

## 3.1 Sketch of Analysis: Reduction to Euclidean Spaces

For space limitations, the proof of Theorem 3.1 is included in the supplemental material, we only provide a sketch of the analysis, carrying the main intuition. The key insight in the proof is that the contribution $\log P(\mathbf{x} \mid \theta)$ to the likelihood $\mathcal{L}(D \mid \theta)$ can be expressed in the following way:

**Lemma 3.2.** *There exist functions $\phi$, $\psi$, and $f$ such that, for any point $\mathbf{x} \in \mathbb{R}^d$ and mixture model $\theta$, $\ln P(\mathbf{x} \mid \theta) = -f_{\phi(\mathbf{x})}(\psi(\theta)) + Z(\theta)$, where*

$$f_{\tilde{\mathbf{x}}}(y) = -\ln \sum_i \tilde{w}_i \exp\left( -W_i \mathrm{dist}(\tilde{\mathbf{x}} - \tilde{\mu}_i, \mathbf{s}_i)^2 \right).$$

*Hereby, $\phi$ is a function that maps a point $\mathbf{x} \in \mathbb{R}^d$ into $\tilde{\mathbf{x}} = \phi(\mathbf{x}) \in \mathbb{R}^{2d}$, and $\psi$ is a function that maps a mixture model $\theta$ into a tuple $y = (s, w, \tilde{\mu}, W)$ where $w$ is a $k$-tuple of nonnegative weights $\tilde{w}_1, \ldots, \tilde{w}_k$ summing to 1, $s = \mathbf{s}_1, \ldots, \mathbf{s}_k \subseteq \mathbb{R}^{2d}$ is a set of $k$ $d$-dimensional subspaces that are weighted by weights $W_1, \cdots, W_k > 0$, and $\tilde{\mu} = \tilde{\mu}_1, \cdots, \tilde{\mu}_k \in \mathbb{R}^{2d}$ is a set of $k$ means.*

The main idea behind Lemma 3.2 is that level sets of distances between points and subspaces are quadratic forms, and can thus represent level sets of the Gaussian probability density function (see Figure 2(a) for an illustration). We recognize the "soft-min" function $\wedge_{\mathbf{w}'}(\eta) \equiv$

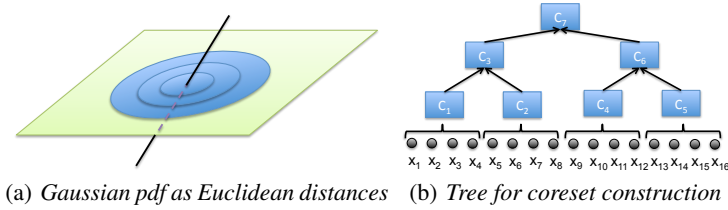

(a) *Gaussian pdf as Euclidean distances*　　(b) *Tree for coreset construction*

Figure 2: (a) Level sets of the distances between points on a plane (green) and (disjoint) $k$-dimensional subspaces are ellipses, and thus can represent contour lines of the multivariate Gaussian. (b) Tree construction for generating coresets in parallel or from data streams. Black arrows indicate "merge-and-compress" operations. The (intermediate) coresets $C_1, \ldots, C_7$ are enumerated in the order in which they would be generated in the streaming case. In the parallel case, $C_1, C_2, C_4$ and $C_5$ would be constructed in parallel, followed by parallel construction of $C_3$ and $C_6$, finally resulting in $C_7$.

$-\ln \sum_i w_i' \exp(-\eta_i)$ as an approximation upper-bounding the minimum $\min(\eta) = \min_i \eta_i$ for $\eta_i = W_i \mathrm{dist}(\tilde{\mathbf{x}} - \tilde{\mu}_i, \mathbf{s_i})^2$ and $\eta = [\eta_1, \ldots, \eta_k]$. The motivation behind this transformation is that it allows expressing the likelihood $P(\mathbf{x} \mid \theta)$ of a data point $\mathbf{x}$ given a model $\theta$ in a purely geometric manner as soft-min over distances between points and subspaces in a transformed space. Notice that if we use the minimum $\min()$ instead of the soft-min $\wedge_{\tilde{w}}()$, we recover the problem of approximating the data set $D$ (transformed via $\phi$) by $k$-subspaces. For semi-spherical Gaussians, it can be shown that the subspaces can be chosen as points while incurring a multiplicative error of at most $1/\varepsilon$, and thus we recover the well-known $k$-means problem in the transformed space. This insight suggests using a known coreset construction for $k$-means, adapted to the transformation employed. The remaining challenge in the proof is to bound the additional error incurred by using the soft-min function $\wedge_{\tilde{w}}(\cdot)$ instead of the minimum $\min(\cdot)$. We tackle this challenge by proving a generalized triangle inequality adapted to the exponential transformation, and employing the framework described in [4], which provides a general method for constructing coresets for clustering problems of the form $\min_{\mathbf{s}} \sum_i f_{\tilde{\mathbf{x}}}(\mathbf{s})$.

As proved in [4], the key quantity that controls the size of a coreset is the pseudo-dimension of the functions $F_d = \{f_{\tilde{\mathbf{x}}} \text{ for } \tilde{\mathbf{x}} \in \mathbb{R}^{2d}\}$. This notion of dimension is closely related to the VC dimension of the (sub-level sets of the) functions $F_d$ and therefore represents the complexity of this set of functions. The final ingredient in the proof of Theorem 3.1 is a new bound on the complexity of mixtures of $k$ Gaussians in $d$ dimensions proved in the supplemental material.

### 3.2 Streaming and Parallel Computation

One major advantage of coresets is that they can be constructed in parallel, as well as in a streaming setting where data points arrive one by one, and it is impossible to remember the entire data set due to memory constraints. The key insight is that coresets satisfy certain composition properties, which have previously been used by [6] for streaming and parallel construction of coresets for geometric clustering problems such as $k$-median and $k$-means.

1. Suppose $C_1$ is a $(k, \varepsilon)$-coreset for $D_1$, and $C_2$ is a $(k, \varepsilon)$-coreset for $D_2$. Then $C_1 \cup C_2$ is a $(k, \varepsilon)$-coreset for $D_1 \cup D_2$.
2. Suppose $C$ is a $(k, \varepsilon)$-coreset for $D$, and $C'$ is a $(k, \delta)$-coreset for $C$. Then $C'$ is a $(k, (1 + \varepsilon)(1 + \delta) - 1)$-coreset for $D$.

In the following, we review how to exploit these properties for parallel and streaming computation.

**Streaming.** In the streaming setting, we assume that points arrive one-by-one, but we do not have enough memory to remember the entire data set. Thus, we wish to maintain a coreset over time, while keeping only a small subset of $\mathcal{O}(\log n)$ coresets in memory. There is a general reduction that shows that a small coreset scheme to a given problem suffices to solve the corresponding problem on a streaming input [7, 6]. The idea is to construct and save in memory a coreset for every block of $\mathrm{poly}(dk/\varepsilon)$ consecutive points arriving in a stream. When we have two coresets in memory, we can merge them (resulting in a $(k, \varepsilon)$-coreset via property (1)), and compress by computing a single coreset from the merged coresets (via property (2)) to avoid increase in the coreset size. An important subtlety arises: While merging two coresets (via property (1)) does not increase the approximation error, compressing a coreset (via property (2)) *does* increase the error. A naive approach that merges and compresses immediately as soon as two coresets have been constructed, can incur an exponential increase in approximation error. Fortunately, it is possible to organize the merge-and-compress operations in a binary tree of height $O(\log n)$, where we need to store in memory a single coreset

for each level on the tree (thus requiring only $\text{poly}(dk\varepsilon^{-1}\log n)$ memory). Figure 2(b) illustrates this tree computation. In order to construct a coreset for the union of two (weighted) coresets, we use a weighted version of Algorithm 1, where we consider a weighted point as duplicate copies of a non-weighted point (possibly with fractional weight). A more formal description can be found in [8]. We summarize our streaming result in the following theorem.

**Theorem 3.3.** *A $(k,\varepsilon)$-coreset for a stream of $n$ points in $\mathbb{R}^d$ can be computed for the $\varepsilon$-semi-spherical GMM problem with probability at least $1 - \delta$ using space and update time $\text{poly}(dk\varepsilon^{-1}\log n\log(1/\delta))$.*

**Parallel/Distributed computations.** Using the same ideas from the streaming model, a (non-parallel) coreset construction can be transformed into a parallel one. We partition the data into sets, and compute coresets for each set, independently, on different computers in a cluster. We then (in parallel) merge (via property (1)) two coresets, and compute a single coreset for every pair of such coresets (via property (2)). Continuing in this manner yields a process that takes $\mathcal{O}(\log n)$ iterations of parallel computation. This computation is also naturally suited for map-reduce [9] style computations, where the map tasks compute coresets for disjoint parts of $D$, and the reduce tasks perform the merge-and-compress operations. Figure 2(b) illustrates this parallel construction.

**Theorem 3.4.** *A $(k,\varepsilon)$-coreset for a set of $n$ points in $\mathbb{R}^d$ can be computed for the $\varepsilon$-semi-spherical GMM problem with probability at least $1 - \delta$ using $m$ machines in time $(n/m) \cdot \text{poly}(dk\varepsilon^{-1}\log(1/\delta))\log n)$.*

### 3.3   Fitting a GMM on the Coreset using Weighted EM

One approach, which we employ in our experiments, is to use a natural generalization of the EM algorithm, which takes the coreset weights into account. We here describe the algorithm for the case of GMMs. For other mixture distributions, the E and M steps are modified appropriately.

---

**Algorithm 2**: Weighted EM for Gaussian mixtures

---
**Input**: Coreset $C$, $k$, TOL
**Output**: Mixture model $\theta_C$
$\mathcal{L}_{old} = \infty$; Initialize means $\mu_1, \ldots, \mu_k$ by sampling $k$ points from $C$ with probability proportional to their weight. Initialize $\Sigma_i = I$ and $w_i = \frac{1}{k}$ for all $i$;
**repeat**

   $\mathcal{L}_{old} = \mathcal{L}(C \mid \theta)$; **for** $j = 1$ **to** $n$ **do**   **for** $i = 1$ **to** $k$ **do**   Compute $\eta_{i,j} = \gamma_i \frac{w_i \mathcal{N}(\mathbf{x}_j'; \mu_i, \Sigma_i)}{\sum_\ell w_\ell \mathcal{N}(\mathbf{x}_j'; \mu_\ell, \Sigma_\ell)}$;

   **for** $i = 1$ **to** $k$ **do**

      $w_i \leftarrow w_i / \sum_\ell w_i$; $\mu_i \leftarrow \sum_j \eta_{i,j} \mathbf{x}_j' / \sum_j \eta_{i,j}$; $\Sigma_i \leftarrow \sum_j \eta_{i,j} (\mathbf{x}_j' - \mu_i)(\mathbf{x}_j' - \mu_i)^T / \sum_j \eta_{i,j}$;
**until** $\mathcal{L}(C \mid \theta) \geq \mathcal{L}_{old} - TOL$ ;

---

Using a similar analysis as for the standard EM algorithm, Algorithm 2 is guaranteed to converge, but only to a local optimum. However, since it is applied on a much smaller set, it can be initialized using multiple random restarts.

## 4   Extensions and Generalizations

We now show how the connection between estimating the parameters for mixture models and problems in computational geometry can be leveraged further. Our observations are based on the link between mixture of Gaussians and projective clustering (multiple subspace approximation) as shown in Lemma 3.2.

**Generalizations to non-semi-spherical GMMs.** For simplicity, we generalized the coreset construction for the $k$-means problem, which required assumptions that the Gaussians are $\varepsilon$-semi-spherical. However, several more complex coresets for projective clustering were suggested recently (*cf.*, [4]). Using the tools developed in this article, each such coreset implies a corresponding coreset for GMMs and generalizations. As an example, the coresets for approximating points by lines [10] implies that we can construct small coresets for GMMs even if the smallest singular value of one of the corresponding covariance matrices is zero.

**Generalizations to $\ell_q$ distances and other norms.** Our analysis is based on combinatorics (such as the complexity of sub-levelsets of GMMs) and probabilistic methods (non-uniform random sampling). Therefore, generalizations to other non-Euclidean distance functions, or error functions such as (non-squared) distances (mixture of Laplace distributions) is straightforward. The main property

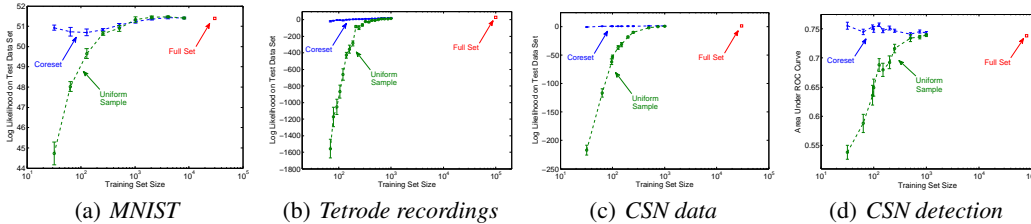

| (a) *MNIST* | (b) *Tetrode recordings* | (c) *CSN data* | (d) *CSN detection* |

Figure 3: Experimental results for three real data sets. We compare likelihood of the best model obtained on subsets $C$ constructed by uniform sampling, and by the adaptive coreset sampling procedure.

that we need is a generalization of the triangle inequality, as proved in the supplemental material. For example, replacing the squared distances by non-squared distances yields a coreset for mixture of Laplace distributions. The double triangle inequality $\|a - c\|^2 \le 2(\|a - b\| + \|b - c\|^2)$ that we used in this paper is replaced by Hölder's inequality, $\|a - c\|^2 \le 2^{O(q)} \|a - b\| + 2 \|b - c\|^2$. Such a result is straight-forward from our analysis, and we summarize it in the following theorem.

**Theorem 4.1.** *Let $q \ge 1$ be an integer. Consider Algorithm 1, where $\mathrm{dist}(\cdot, \cdot)^2$ is replaced by $\mathrm{dist}(\cdot, \cdot)^q$ and $\varepsilon^2$ is replaced by $\varepsilon^{O(q)}$. Suppose $C$ is sampled from $D$ using this updated version of Algorithm 1 for parameters $\varepsilon, \delta$ and $k$. Then, with prob. at least $1 - \delta$ it holds that for all $\theta \in \mathfrak{C}_\varepsilon$,*

$$\phi(D \mid \theta)(1 - \varepsilon) \le \phi(C \mid \theta) \le \phi(D \mid \theta)(1 + \varepsilon),$$

*where $Z(\theta) = \sum_i \frac{w_i}{g(\theta_i)}$ and $\phi(D \mid \theta) = -\sum_{\mathbf{x} \in D} \ln \sum_{i=1}^k \frac{w_i}{Z(\theta)g(\theta_i)} \exp\left(-\frac{1}{2}\left\|\Sigma_i^{-1/2}(\mathbf{x} - \mu_i)\right\|^q\right)$ using the normalizer $g(\theta_i) = \int \exp\left(-\frac{1}{2}\left\|\Sigma_i^{-1/2}(\mathbf{x} - \mu_i)\right\|^q\right) d\mathbf{x}$.*

## 5 Experiments

We experimentally evaluate the effectiveness of using coresets of different sizes for training mixture models. We compare against running EM on the full set, as well as on an unweighted, uniform sample from $D$. Results are presented for three real datasets.

**MNIST handwritten digits.** The MNIST dataset contains 60,000 training and 10,000 testing grayscale images of handwritten digits. As in [11], we normalize each component of the data to have zero mean and unit variance, and then reduce each 784-pixel (28x28) image using PCA, retaining only the top $d = 100$ principal components as a feature vector. From the training set, we produce coresets and uniformly sampled subsets of sizes between 30 and 5000, using the parameters $k = 10$ (a cluster for each digit), $\beta = 20$ and $\delta = 0.1$ (see Algorithm 1), and fit GMMs using EM with 3 random restarts. The log likelihood (LLH) of each model on the testing data is shown in Figure 3(a). Notice that coresets significantly outperform uniform samples of the same size, and even a coreset of 30 points performs very well. Further note how the test-log likelihood begins to flatten out for $|C| = 1000$. Constructing the coreset and running EM on this size takes 7.9 seconds (Intel Xeon 2.6 GHz), over 100 times faster than running EM on the full set (15 minutes).

**Neural tetrode recordings.** We also compare coresets and uniform sampling on a large dataset containing 319,209 records of rat hippocampal action potentials, measured by four co-located electrodes. As done by [11], we concatenate the 38-sample waveforms produced by each electrode to obtain a 152-dimensional vector. The vectors are normalized so each component has zero mean and unit variance. The 319,209 records are divided in half to obtain training and testing sets. From the training set, we produce coresets and uniformly sampled subsets of sizes between 70 and 1000, using the parameters $k = 33$ (as in [11]), $\beta = 66$, and $\delta = 0.1$, and fit GMMs. The log likelihood of each model on the held-out testing data is shown in Figure 3(b). Coreset GMMs obtain consistently higher LLH than uniform sample GMMs for sets of the same size, and even a coreset of 100 points performs very well. Overall, training on coresets achieves approximately the same likelihood as training on the full set about 95 times faster (1.2 minutes vs. 1.9 hours).

**CSN cell phone accelerometer data.** Smart phones with accelerometers are being used by the Community Seismic Network (CSN) as inexpensive seismometers for earthquake detection. In [12], 7 GB of acceleration data were recorded from volunteers while carrying and operating their phone in normal conditions (walking, talking, on desk, etc.). From this data, 17-dimensional feature vectors were computed (containing frequency information, moments, etc.). The goal is to train, in an online

fashion, GMMs based on normal data, which then can be used to perform anomaly detection to detect possible seismic activity. Motivated by the limited storage on smart phones, we evaluate coresets on a data set of 40,000 accelerometer feature vectors, using the parameters $k = 6$, $\beta = 12$, and $\delta = 0.1$. Figure 3(c) presents the results of this experiment. Notice that on this data set, coresets show an even larger improvement over uniform sampling. We hypothesize that this is due to the fact that the recorded accelerometer data is imbalanced, and contains clusters of vastly varying size, so uniform sampling does not represent smaller clusters well. Overall, the coresets obtain a speedup of approximately 35 compared to training on the full set. We also evaluate how GMMs trained on the coreset compare with the baseline GMMs in terms of anomaly detection performance. For each GMM, we compute ROC curves measuring the performance of detecting earthquake recordings from the Southern California Seismic Network (cf., [12]). Note that even very small coresets lead to performance comparable to training on the full set, drastically outperforming uniform sampling (Fig. 3(d)).

## 6 Related Work

**Theoretical results on mixtures of Gaussians.** There has been a significant amount of work on learning and applying GMMs (and more general distributions). Perhaps the most commonly used technique in practice is the EM algorithm [13], which is however only guaranteed to converge to a local optimum of the likelihood. Dasgupta [14] is the first to show that parameters of an unknown GMM $P$ can be estimated in polynomial time, with arbitrary accuracy $\varepsilon$, given i.i.d. samples from $P$. However, his algorithm assumes a common covariance, bounded excentricity, a (known) bound on the smallest component weight, as well as a separation (distance of the means), that scales as $\Omega(\sqrt{d})$. Subsequent works relax the assumption on separation to $d^{1/4}$ [15] and $k^{1/4}$ [16]. [3] is the first to learn general GMMs, with separation $d^{1/4}$. [17] provides the first result that does not require any separation, but assumes that the Gaussians are axis-aligned. Recently, [18] and [19] provide algorithms with polynomial running time (except exponential dependence on $k$) and sample complexity for arbitrary GMMs. However, in contrast to our results, all the results described above crucially rely on the fact that the data set $D$ is actually generated by a mixture of Gaussians. The problem of fitting a mixture model with near-optimal log-likelihood for arbitrary data is studied by [3], who provides a PTAS for this problem. However, their result requires that the Gaussians are identical spheres, in which case the maximum likelihood problem is identical to the k-means problem. In contrast, our results make only mild assumptions about the Gaussian components. Furthermore, none of the algorithms described above applies to the streaming or parallel setting.

**Coresets.** Approximation algorithms in computational geometry often make use of random sampling, feature extraction, and $\epsilon$-samples [20]. Coresets can be viewed as a general concept that includes all of the above, and more. See a comprehensive survey on this topic in [4]. It is not clear that there is any commonly agreed-upon definition of a coreset, despite several inconsistent attempts to do so [6, 8]. Coresets have been the subject of many recent papers and several surveys [1, 2]. They have been used to great effect for a host of geometric and graph problems, including $k$-median [6], $k$-mean [8], $k$-center [21], $k$-line median [10] subspace approximation [10, 22], etc. Coresets also imply streaming algorithms for many of these problems [6, 1, 23, 8]. A framework that generalizes and improves several of these results has recently appeared in [4].

## 7 Conclusion

We have shown how to construct coresets for estimating parameters of GMMs and natural generalizations. Our construction hinges on a natural connection between statistical estimation and clustering problems in computational geometry. To our knowledge, our results provide the first rigorous guarantees for obtaining compressed $\varepsilon$-approximations of the log-likelihood of mixture models for large data sets. The coreset construction relies on an intuitive adaptive sampling scheme, and can be easily implemented. By exploiting certain closure properties of coresets, it is possible to construct them in parallel, or in a single pass through a stream of data, using only $\text{poly}(dk\varepsilon^{-1} \log n \log(1/\delta))$ space and update time. Unlike most of the related work, our coresets provide guarantees for any given (possibly unstructured) data, without assumptions on the distribution or model that generated it. Lastly, we apply our construction on three real data sets, demonstrating significant gains over no or naive subsampling.

**Acknowledgments** This research was partially supported by ONR grant N00014-09-1-1044, NSF grants CNS-0932392, IIS-0953413 and DARPA MSEE grant FA8650-11-1-7156.

## Footnotes

[1]equivalently, $\mathfrak{C}$ can be interpreted as prior thresholding.

# References

[1] P. K. Agarwal, S. Har-Peled, and K. R. Varadarajan. Geometric approximations via coresets. *Combinatorial and Computational Geometry - MSRI Publications*, 52:1–30, 2005.

[2] A. Czumaj and C. Sohler. Sublinear-time approximation algorithms for clustering via random sampling. *Random Struct. Algorithms (RSA)*, 30(1-2):226–256, 2007.

[3] Sanjeev Arora and Ravi Kannan. Learning mixtures of separated nonspherical gaussians. *Annals of Applied Probability*, 15(1A):69–92, 2005.

[4] D. Feldman and M. Langberg. A unified framework for approximating and clustering data. In *Proc. 41th Annu. ACM Symp. on Theory of Computing (STOC)*, 2011.

[5] S. Har-Peled and A. Kushal. Smaller coresets for $k$-median and $k$-means clustering. *Discrete & Computational Geometry*, 37(1):3–19, 2007.

[6] S. Har-Peled and S. Mazumdar. On coresets for k-means and k-median clustering. In *Proc. 36th Annu. ACM Symp. on Theory of Computing (STOC)*, pages 291–300, 2004.

[7] Jon Louis Bentley and James B. Saxe. Decomposable searching problems i: Static-to-dynamic transformation. *J. Algorithms*, 1(4):301–358, 1980.

[8] D. Feldman, M. Monemizadeh, and C. Sohler. A PTAS for k-means clustering based on weak coresets. In *Proc. 23rd ACM Symp. on Computational Geometry (SoCG)*, pages 11–18, 2007.

[9] Jeffrey Dean and Sanjay Ghemawat. Mapreduce: Simplified data processing on large clusters. In *OSDI'04: Sixth Symposium on Operating System Design and Implementation*, 2004.

[10] D. Feldman, A. Fiat, and M. Sharir. Coresets for weighted facilities and their applications. In *Proc. 47th IEEE Annu. Symp. on Foundations of Computer Science (FOCS)*, pages 315–324, 2006.

[11] Ryan Gomes, Andreas Krause, and Pietro Perona. Discriminative clustering by regularized information maximization. In *Proc. Neural Information Processing Systems (NIPS)*, 2010.

[12] Matthew Faulkner, Michael Olson, Rishi Chandy, Jonathan Krause, K. Mani Chandy, and Andreas Krause. The next big one: Detecting earthquakes and other rare events from community-based sensors. In *In Proc. ACM/IEEE International Conference on Information Processing in Sensor Networks (IPSN)*, 2011.

[13] A. P. Dempster, N. M. Laird, and D. B. Rubin. Maximum likelihood from incomplete data via the em algorithm. *J. Roy. Statist. Soc. Ser. B*, 39:1–38, 1977.

[14] S. Dasgupta. Learning mixtures of gaussians. In *Fortieth Annual IEEE Symposium on Foundations of Computer Science (FOCS)*, 1999.

[15] S. Dasgupta and L.J. Schulman. A two-round variant of em for gaussian mixtures. In *Sixteenth Conference on Uncertainty in Artificial Intelligence (UAI)*, 2000.

[16] S. Vempala and G. Wang. A spectral algorithm for learning mixture models. In *In Proceedings of the 43rd Annual IEEE Symposium on Foundations of Computer Science*, 2002.

[17] J. Feldman, R. A. Servedio, and R. O'Donnell. Pac learning axis-aligned mixtures of gaussians with no separation assumption. In *COLT*, 2006.

[18] A. Moitra and G. Valiant. Settling the polynomial learnability of mixtures of gaussians. In *In Proc. Foundations of Computer Science (FOCS)*, 2010.

[19] M. Belkin and K. Sinha. Polynomial learning of distribution families. In *In Proc. Foundations of Computer Science (FOCS)*, 2010.

[20] D. Haussler. Decision theoretic generalizations of the PAC model for neural net and other learning applications. *Inf. Comput.*, 100(1):78–150, 1992.

[21] S. Har-Peled and K. R. Varadarajan. High-dimensional shape fitting in linear time. *Discrete & Computational Geometry*, 32(2):269–288, 2004.

[22] M.W. Mahoney and P. Drineas. CUR matrix decompositions for improved data analysis. *Proceedings of the National Academy of Sciences*, 106(3):697, 2009.

[23] G. Frahling and C. Sohler. Coresets in dynamic geometric data streams. In *Proc. 37th Annu. ACM Symp. on Theory of Computing (STOC)*, pages 209–217, 2005.

